# Synergistic Face Detection and Pose Estimation with Energy-Based Models

**Margarita Osadchy**
NEC Labs America
Princeton NJ 08540
rita@osadchy.net

**Matthew L. Miller**
NEC Labs America
Princeton NJ 08540
mlm@nec-labs.com

**Yann Le Cun**
The Courant Institute
New York University
yann@cs.nyu.edu

## Abstract

We describe a novel method for real-time, simultaneous multi-view face detection and facial pose estimation. The method employs a convolutional network to map face images to points on a manifold, parametrized by pose, and non-face images to points far from that manifold. This network is trained by optimizing a loss function of three variables: image, pose, and face/non-face label. We test the resulting system, in a single configuration, on three standard data sets – one for frontal pose, one for rotated faces, and one for profiles – and find that its performance on each set is comparable to previous multi-view face detectors that can only handle one form of pose variation. We also show experimentally that the system's accuracy on *both* face detection and pose estimation is improved by training for the two tasks together.

## 1   Introduction

The detection of human faces in natural images and videos is a key component in a wide variety of applications of human-computer interaction, search and indexing, security, and surveillance. Many real-world applications would profit from *multi-view* detectors that can detect faces under a wide range of poses: looking left or right (yaw axis), up or down (pitch axis), or tilting left or right (roll axis).

In this paper we describe a novel method that not only detects faces independently of their poses, but simultaneously estimates those poses. The system is highly-reliable, runs at near real time (5 frames per second on standard hardware), and is robust against variations in yaw ($\pm 90°$), roll ($\pm 45°$), and pitch ($\pm 60°$).

The method is motivated by the idea that multi-view face detection and pose estimation are so closely related that they should not be performed separately. The tasks are related in the sense that they must be robust against the same sorts of variation: skin color, glasses, facial hair, lighting, scale, expressions, etc. We suspect that, when trained together, each task can serve as an inductive bias for the other, yielding better generalization or requiring fewer training examples [2].

To exploit the synergy between these two tasks, we train a convolutional network to map face images to points on a *face manifold*, and non-face images to points far away from that manifold. The manifold is parameterized by facial pose. Conceptually, we can view the pose parameter as a latent variable that can be inferred through an *energy-minimization process* [4]. To train the machine we derive a new type of *discriminative loss function* that is tailored to such detection tasks.

**Previous Work:** Learning-based approaches to face detection abound, including real-time methods [16], and approaches based on convolutional networks [15, 3]. Most multi-view systems take a *view-based* approach, which involves building separate detectors for different views and either applying them in parallel [10, 14, 13, 7] or using a pose estimator to select a detector [5]. Another approach is to estimate and correct in-plane rotations before applying a single pose-specific detector [12]. Closer to our approach is that of [8], in which a number of Support Vector Regressors are trained to approximate smooth functions, each of which has a maximum for a face at a particular pose. Another machine is trained to convert the resulting values to estimates of poses, and a third is trained to convert the values into a face/non-face score. The resulting system is very slow.

## 2 Integrating face detection and pose estimation

To exploit the posited synergy between face detection and pose estimation, we must design a system that integrates the solutions to the two problems. We hope to obtain better results on *both* tasks, so this should not be a mere cascaded system in which the answer to one problem is used to assist in solving the other. Both answers must be derived from one underlying analysis of the input, and both tasks must be trained together.

Our approach is to build a trainable system that can map raw images $X$ to points in a low-dimensional space. In that space, we pre-define a *face manifold* $F(Z)$ that we parameterize by the pose $Z$. We train the system to map face images with known poses to the corresponding points on the manifold. We also train it to map non-face images to points far away from the manifold. Proximity to the manifold then tells us whether or not an image is a face, and projection to the manifold yields an estimate of the pose.

**Parameterizing the Face Manifold**: We will now describe the details of the parameterizations of the face manifold. Let's start with the simplest case of one pose parameter $Z = \theta$, representing, say, yaw. If we want to preserve the natural topology and geometry of the problem, the face manifold under yaw variations in the interval $[-90°, 90°]$ should be a half circle (with constant curvature). We embed this half-circle in a three-dimensional space using three equally-spaced shifted cosines.

$$F_i(\theta) = \cos(\theta - \alpha_i); \quad i = 1, 2, 3; \quad \theta = [-\frac{\pi}{2}, \frac{\pi}{2}] \tag{1}$$

When we run the network on an image $X$, it outputs a vector $G(X)$ with three components that can be decoded analytically into corresponding pose angle:

$$\overline{\theta} = \arctan \frac{\sum_{i=1}^{3} G_i(X) \cos(\alpha_i)}{\sum_{i=1}^{3} G_i(X) \sin(\alpha_i)} \tag{2}$$

The point on the manifold closest to $G(X)$ is just $F(\overline{\theta})$.

The same idea can be applied to any number of pose parameters. Let us consider the set of all faces with yaw in $[-90, 90]$ and roll in $[-45, 45]$. In an abstract way, this set is isomorphic to a portion of the surface of a sphere. Consequently, we encode the pose with the product of the cosines of the two angles:

$$F_{ij}(\theta, \phi) = \cos(\theta - \alpha_i) \cos(\phi - \beta_j); \quad i, j = 1, 2, 3; \tag{3}$$

For convenience we rescale the roll angles to the range of $[-90, 90]$. With these parameterizations, the manifold has constant curvature, which ensures that the effect of errors will be the same regardless of pose. Given nine components of the network's output $G_{ij}(X)$, we compute the corresponding pose angles as follows:

$$
\begin{aligned}
cc &= \sum_{ij} G_{ij}(X) \cos(\alpha_i) \cos(\beta_j); \quad cs = \sum_{ij} G_{ij}(X) \cos(\alpha_i) \sin(\beta_j) \\
sc &= \sum_{ij} G_{ij}(X) \sin(\alpha_i) \cos(\beta_j); \quad ss = \sum_{ij} G_{ij}(X) \sin(\alpha_i) \sin(\beta_j) \\
\overline{\theta} &= 0.5(atan2(cs + sc, cc - ss) + atan2(sc - cs, cc + ss)) \\
\overline{\phi} &= 0.5(atan2(cs + sc, cc - ss) - atan2(sc - cs, cc + ss))
\end{aligned}
\tag{4}
$$

Note that the dimension of the face manifold is much lower than that of the embedding space. This gives ample space to represent non-faces away from the manifold.

## 3   Learning Machine

To build a learning machine for the proposed approach we refer to the *Minimum Energy Machine* framework described in [4].

**Energy Minimization Framework**: We can view our system as a scalar-value function $E_W(Y, Z, X)$, where $X$ and $Z$ are as defined above, $Y$ is a binary label ($Y = 1$ for face, $Y = 0$ for a non-face), and $W$ is a parameter vector subject to learning. $E_W(Y, Z, X)$ can be interpreted as an *energy function* that measures the degree of compatibility between $X, Z, Y$. If $X$ is a face with pose $Z$, then we want: $E_W(1, Z, X) \ll E_W(0, Z', X)$ for any pose $Z'$, and $E_W(1, Z', X) \gg E_W(1, Z, X)$ for any pose $Z' \neq Z$.

Operating the machine consists in clamping $X$ to the observed value (the image), and finding the values of $Z$ and $Y$ that minimize $E_W(Y, Z, X)$:

$$(\overline{Y}, \overline{Z}) = \mathrm{argmin}_{Y \in \{Y\},\, Z \in \{Z\}} E_W(Y, Z, X) \tag{5}$$

where $\{Y\} = \{0, 1\}$ and $\{Z\} = [-90, 90] \times [-45, 45]$ for yaw and roll variables. Although this inference process can be viewed probabilistically as finding the most likely configuration of $Y$ and $Z$ according to a model that attributes high probabilities to low-energy configurations (e.g. a Gibbs distribution), we view it as a non probabilistic decision making process. In other words, we make no assumption as to the finiteness of integrals over $\{Y\}$ and $\{Z\}$ that would be necessary for a properly normalized probabilistic model. This gives us considerable flexibility in the choice of the internal architecture of $E_W(Y, Z, X)$.

Our energy function for a face $E_W(1, Z, X)$ is defined as the distance between the point produced by the network $G_W(X)$ and the point with pose $Z$ on the manifold $F(Z)$:

$$E_W(1, Z, X) = \|G_W(X) - F(Z)\| \tag{6}$$

The energy function for a non-face $E_W(0, Z, X)$ is equal to a constant $T$ that we can interpret as a threshold (it is independent of $Z$ and $X$). The complete energy function is:

$$E_W(Y, Z, X) = Y\|G_W(X) - F(Z)\| + (1 - Y)T \tag{7}$$

The architecture of the machine is depicted in Figure 1. Operating this machine (finding the output label and pose with the smallest energy) comes down to first finding: $\overline{Z} = \mathrm{argmin}_{Z \in \{Z\}} \|G_W(X) - F(Z)\|$, and then comparing this minimum distance, $\|G_W(X) - F(\overline{Z})\|$, to the threshold $T$. If it smaller than $T$, then $X$ is classified as a face, otherwise $X$ is classified as a non-face. This decision is implemented in the architecture as a *switch*, that depends upon the binary variable $Y$.

**Convolutional Network**: We employ a Convolutional Network as the basic architecture for the $G_W(X)$ image-to-face-space mapping function. Convolutional networks [6] are "end-to-end" trainable system that can operate on raw pixel images and learn low-level features and high-level representation in an integrated fashion. Convolutional nets are advantageous because they easily learn the types of shift-invariant local features that are relevant to image recognition; and more importantly, they can be replicated over large images (swept over every location) at a fraction of the cost of replicating more traditional classifiers [6]. This is a *considerable advantage for building real-time systems*.

We employ a network architecture similar to LeNet5 [6]. The difference is in the number of maps. In our architecture we have 8 feature maps in the bottom convolutional and subsampling layers and 20 maps in the next two layers. The last layer has 9 outputs to encode two pose parameters.

**Training with a Discriminative Loss Function for Detection**: We define the loss function as follows:

$$\mathcal{L}(W) = \frac{1}{|\mathcal{S}_1|} \sum_{i \in \mathcal{S}_1} L_1(W, Z^i, X^i) + \frac{1}{|\mathcal{S}_0|} \sum_{i \in \mathcal{S}_0} L_0(W, X^i) \tag{8}$$

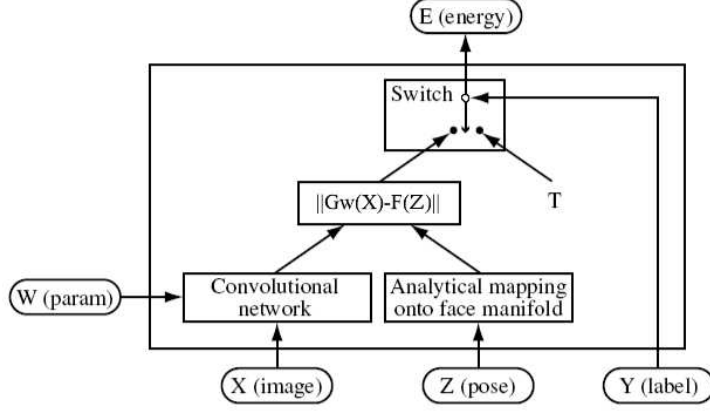

Figure 1: *Architecture of the Minimum Energy Machine.*

where $\mathcal{S}_1$ is the set of training faces, $\mathcal{S}_0$ the set of non-faces, $L_1(W, Z^i, X^i)$ and $L_0(W, X^i)$ are loss functions for a face sample (with a known pose) and non-face, respectively[1].

The loss $\mathcal{L}(W)$ should be designed so that its minimization for a particular positive training sample $(X^i, Z^i, 1)$, will make $E_W(1, Z^i, X^i) < E_W(Y, Z, X^i)$ for $Y \neq Y^i$ or $Z \neq Z^i$. To satisfy this, it is sufficient to make $E_W(1, Z^i, X^i) < E_W(0, \overline{Z}, X^i)$. For a particular negative training sample $(X^i, 0)$, minimizing the loss should make $E_W(1, Z, X^i) > E_W(0, Z, X^i) = T$ for any $Z$. To satisfy this, it is sufficient to make $E_W(1, \overline{Z}, X^i) > T$.

Let $W$ be the current parameter value, and $W'$ be the parameter value after an update caused by a single sample. To cause the machine to achieve the desired behavior, we need the parameter update to decrease the difference between the energy of the desired label and the energy of the undesired label. In our case, since $E_W(0, Z, X) = T$ is constant, the following condition on the update is sufficient to ensure the desired behavior:

**Condition 1.** *for a face example $(X, Z, 1)$, we must have: $E_{W'}(1, Z, X) < E_W(1, Z, X)$ For a non-face example $(X, 1)$, we must have: $E_{W'}(1, \overline{Z}, X) > E_W(1, \overline{Z}, X)$*

We choose the following forms for $L_1$ and $L_0$:

$$L_1(W, 1, Z, X) = E_W(1, Z, X)^2; \quad L_0(W, 0, X) = K \exp[-E(1, \overline{Z}, X)] \quad (9)$$

where $K$ is a positive constant.

Next we show that minimizing (9) with an incremental gradient-based algorithm will satisfy condition 1. With gradient-based optimization algorithms, the parameter update formula is of the form: $\delta W = W' - W = -\eta A \frac{\partial L}{\partial W}$. where $A$ is a judiciously chosen symmetric positive semi-definite matrix, and $\eta$ is a small positive constant.

For $Y = 1$ (face): An update step will change the parameter by $\delta W = -\eta A \frac{\partial E_W(1,Z,X)^2}{\partial W} = -2\eta E_W(1, Z, X) A \frac{\partial E_W(1,Z,X)}{\partial W}$. To first order (for small values of $\eta$), the resulting change in $E_W(1, Z, X)$ is given by:

$$\frac{\partial E_W(1, Z, X)}{\partial W}^T \delta W = -2\eta E_W(1, Z, X) \frac{\partial E_W(1, Z, X)}{\partial W}^T A \frac{\partial E_W(1, Z, X)}{\partial W} < 0$$

because $E_W(1, Z, X) > 0$ (it's a distance), and the quadratic form is positive. Therefore $E_{W'}(1, Z, X) < E_W(1, Z, X)$.

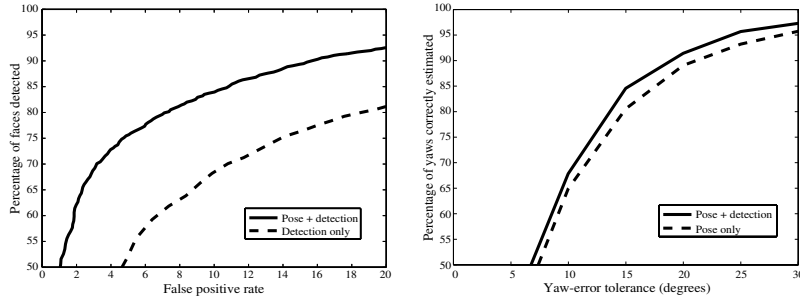

Figure 2: *Synergy test. Left: ROC curves for the pose-plus-detection and detection-only networks. Right: frequency with which the pose-plus-detection and pose-only networks correctly estimated the yaws within various error tolerances.*

For $Y = 0$ (non-face): An update step will change the parameter by $\delta W = -\eta A \frac{\partial K \exp[-E(1,\overline{Z},X)]}{\partial W} = \eta K \exp[-E_W(1,\overline{Z},X)] \frac{\partial E_W(1,\overline{Z},X)}{\partial W}$. To first order (for small values of $\eta$), the resulting change in $E_W(1,\overline{Z},X)$ is given by:

$$\frac{\partial E_W(1,Z,X)}{\partial W}^T \delta W = \eta K \exp[-E_W(1,\overline{Z},X)] \; \frac{\partial E_W(1,\overline{Z},X)}{\partial W}^T A \frac{\partial E_W(1,\overline{Z},X)}{\partial W} > 0$$

Therefore $E_{W'}(1,\overline{Z},X) > E_W(1,\overline{Z},X)$.

**Running the Machine:** Our detection system works on grayscale images and it applies the network to each image at a range of scales, stepping by a factor of $\sqrt{2}$. The network is replicated over the image at each scale, stepping by 4 pixels in x and y (this step size is a consequence of having two, 2x2 subsampling layers). At each scale and location, the network outputs are compared to the closest point on the manifold, and the system collects a list of all instances closer than our detection threshold. Finally, after examining all scales, the system identifies groups of overlapping detections in the list and discards all but the strongest (closest to the manifold) from each group. No attempt is made to combine detections or apply any voting scheme. We have implemented the system in C. The system can detect, locate, and estimate the pose of faces that are between 40 and 250 pixels high in a $640 \times 480$ image at roughly 5 frames per second on a 2.4GHz Pentium 4.

## 4 Experiments and results

Using the above architecture, we built a detector to locate faces and estimate two pose parameters: yaw from left to right profile, and in-plane rotation from $-45$ to $45$ degrees. The machine was trained to be robust against pitch variation.

In this section, we first describe the training regimen for this network, and then give the results of two sets of experiments. The first set of experiments tests whether training for the two tasks together improves performance on both. The second set allows comparisons between our system and other published multi-view detectors.

**Training:** Our training set consisted of $52,850$, 32x32-pixel faces from natural images collected at NEC Labs and hand annotated with appropriate facial poses (see [9] for a description of how the annotation was done). These faces were selected from a much larger annotated set to yield a roughly uniform distribution of poses from left profile to right profile, with as much variation in pitch as we could obtain. Our initial negative training data consisted of $52,850$ image patches chosen randomly from non-face areas of a variety of images. For our second set of tests, we replaced half of these with image patches obtained by running the initial version of the detector on our training images and collecting false detections. Each training image was used 5 times during training, with random variations

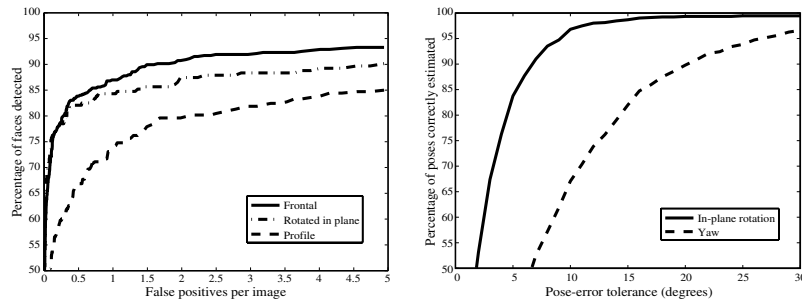

Figure 3: *Results on standard data sets. Left: ROC curves for our detector on the three data sets. The x axis is the average number of false positives per image over all three sets, so each point corresponds to a single detection threshold. Right: frequency with which yaw and roll are estimated within various error tolerances.*

in scale (from $x\sqrt{2}$ to $x(1 + \sqrt{2})$), in-plane rotation ($\pm45°$), brightness ($\pm20$), contrast (from 0.8 to 1.3).

To train the network, we made 9 passes through this data, though it mostly converged after about the first 6 passes. Training was performed using LUSH [1], and the total training time was about 26 hours on a 2Ghz Pentium 4. At the end of training, the network had converged to an equal error rate of 5% on the training data and 6% on a separate test set of 90,000 images.

**Synergy tests:** The goal of the synergy test was to verify that both face detection and pose estimation benefit from learning and running in parallel. To test this claim we built three networks with almost identical architectures, but trained to perform different tasks. The first one was trained for simultaneous face detection and pose estimation (combined), the second was trained for detection only and the third for pose estimation only. The "detection only" network had only one output for indicating whether or not its input was a face. The "pose only" network was identical to the combined network, but trained on faces only (no negative examples). Figure 2 shows the results of running these networks on our 10,000 test images. In both these graphs, we see that the pose-plus-detection network had better performance, confirming that training for each task benefits the other.

**Standard data sets:** There is no standard data set that tests all the poses our system is designed to detect. There are, however, data sets that have been used to test more restricted face detectors, each set focusing on a particular variation in pose. By testing a single detector with all of these sets, we can compare our performance against published systems. As far as we know, we are the first to publish results for a single detector on all these data sets. The details of these sets are described below:
• MIT+CMU [14, 11] – 130 images for testing frontal face detectors. We count 517 faces in this set, but the standard tests only use a subset of 507 faces, because 10 faces are in the wrong pose or otherwise not suitable for the test. (Note: about 2% of the faces in the standard subset are badly-drawn cartoons, which we do not intend our system to detect. Nevertheless, we include them in the results we report.)
• TILTED [12] – 50 images of frontal faces with in-plane rotations. 223 faces out of 225 are in the standard subset. (Note: about 20% of the faces in the standard subset are outside of the $\pm45°$ rotation range for which our system is designed. Again, we still include these in our results.)
• PROFILE [13] – 208 images of faces in profile. There seems to be some disagreement about the number of faces in the standard set of annotations: [13] reports using 347 faces of the 462 that we found, [5] reports using 355, and we found 353 annotations. However, these discrepencies should not significantly effect the reported results.

We counted a face as being detected if 1) at least one detection lay within a circle centered on the midpoint between the eyes, with a radius equal to 1.25 times the distance from that point to the midpoint of the mouth, and 2) that detection came at a scale within a factor of

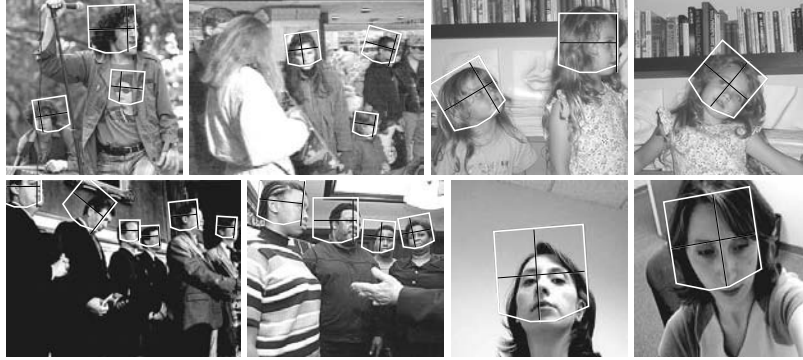

Figure 4: *Some example face detections. Each white box shows the location of a detected face. The angle of each box indicates the estimated in-plane rotation. The black crosshairs within each box indicate the estimated yaw.*

| Data set → | TILTED | | PROFILE | | MIT+CMU | |
|---|---|---|---|---|---|---|
| *False positives per image →* | 4.42 | 26.90 | .47 | 3.36 | .50 | 1.28 |
| **Our detector** | **90%** | **97%** | **67%** | **83%** | **83%** | **88%** |
| **Jones & Viola [5] (tilted)** | **90%** | **95%** | x | | x | |
| **Jones & Viola [5] (profile)** | x | | **70%** | **83%** | x | |
| Rowley *et al* [11] | 89% | 96% | x | | x | |
| Schneiderman & Kanade [13] | x | | 86% | 93% | x | |

Table 1: *Comparisons of our results with other multi-view detectors. Each column shows the detection rates for a given average number of false positives per image (these rates correspond to those for which other authors have reported results). Results for real-time detectors are shown in bold. Note that ours is the only single detector that can be tested on all data sets simultaneously.*

two of the correct scale for the face's size. We counted a detection as a false positive if it did not lie within this range for any of the faces in the image, including those faces not in the standard subset.

The left graph in Figure 3 shows ROC curves for our detector on the three data sets. Figure 4 shows a few results on various poses. Table 1 shows our detection rates compared against other systems for which results were given on these data sets. The table shows that our results on the TILTED and PROFILE sets are similar to those of the two Jones & Viola detectors, and even approach those of the Rowley *et al* and Schneiderman & Kanade non-real-time detectors. Those detectors, however, are not designed to handle all variations in pose, and do not yield pose estimates.

The right side of Figure 3 shows our performance at pose estimation. To make this graph, we fixed the detection threshold at a value that resulted in about 0.5 false positives per image over all three data sets. We then compared the pose estimates for all detected faces (including those not in the standard subsets) against our manual pose annotations. Note that this test is more difficult than typical tests of pose estimation systems, where faces are first localized by hand. When we hand-localize these faces, 89% of yaws and 100% of in-plane rotations are correctly estimated to within $15°$.

## 5 Conclusion

The system we have presented here integrates detection and pose estimation by training a convolutional network to map faces to points on a manifold, parameterized by pose, and non-faces to points far from the manifold. The network is trained by optimizing a loss function of three variables – image, pose, and face/non-face label. When the three variables match, the energy function is trained to have a small value, when they do not match, it is

trained to have a large value.

This system has several desirable properties:
• The use of a convolutional network makes it fast. At typical webcam resolutions, it can process 5 frames per second on a 2.4Ghz Pentium 4.
• It is robust to a wide range of poses, including variations in yaw up to $\pm90°$, in-plane rotation up to $\pm45°$, and pitch up to $\pm60°$. This has been verified with tests on three standard data sets, each designed to test robustness against a single dimension of pose variation.
• At the same time that it detects faces, it produces estimates of their pose. On the standard data sets, the estimates of yaw and in-plane rotation are within $15°$ of manual estimates over 80% and 95% of the time, respectively.

We have shown experimentally that our system's accuracy at *both* pose estimation and face detection is increased by training for the two tasks together.

## Footnotes

[1] Although face samples whose pose is unknown can easily be accommodated, we will not discuss this possibility here.

# References

[1] L. Bottou and Y. LeCun. *The Lush Manual.* `http://lush.sf.net`, 2002.

[2] R. Caruana. Multitask learning. *Machine Learning*, 28:41–75, 1997.

[3] C. Garcia and M. Delakis. A neural architecture for fast and robust face detection. *IEEE-IAPR Int. Conference on Pattern Recognition*, pages 40–43, 2002.

[4] F. J. Huang and Y. LeCun. Loss functions for discriminative training of energy-based graphical models. Technical report, Courant Institute of Mathematical Science, NYU, June 2004.

[5] M. Jones and P. Viola. Fast multi-view face detection. Technical Report TR2003-96, Mitsubishi Electric Research Laboratories, 2003.

[6] Y. LeCun, L. Bottou, Y. Bengio, and P. Haffner. Gradient-based learning applied to document recognition. *Proceedings of the IEEE*, 86(11):2278–2324, November 1998.

[7] S. Z. Li, L. Zhu, Z. Zhang, A. Blake, H. Zhang, and H. Shum. Statistical learning of multi-view face detection. In *Proceedings of the 7th European Conference on Computer Vision-Part IV*, 2002.

[8] Y. Li, S. Gong, and H. Liddell. Support vector regression and classification based multi-view face detection and recognition. In *Face and Gesture*, 2000.

[9] H. Moon and M. L. Miller. Estimating facial pose from sparse representation. In *International Conference on Image Processing*, Singapore, 2004.

[10] A. Pentland, B. Moghaddam, and T. Starner. View-based and modular eigenspaces for face recognition. In *CVPR*, 1994.

[11] H. A. Rowley, S. Baluja, and T. Kanade. Neural network-based face detection. *PAMI*, 20:22–38, 1998.

[12] H. A. Rowley, S. Baluja, and T. Kanade. Rotation invariant neural network-based face detection. In *Computer Vision and Pattern Recognition*, 1998.

[13] H. Schneidermn and T. Kanade. A statistical method for 3d object detection applied to faces and cars. In *Computer Vision and Pattern Recognition*, 2000.

[14] K. Sung and T. Poggio. Example-based learning of view-based human face detection. *PAMI*, 20:39–51, 1998.

[15] R. Vaillant, C. Monrocq, and Y. LeCun. Original approach for the localisation of objects in images. *IEE Proc on Vision, Image, and Signal Processing*, 141(4):245–250, August 1994.

[16] P. Viola and M. Jones. Rapid object detection using a boosted cascade of simple features. In *Proceedings IEEE Conf. on Computer Vision and Pattern Recognition*, pages 511–518, 2001.
